# Analysis of Unstandardized Contributions in Cross Connected Networks

**Thomas R. Shultz**
shultz@psych.mcgill.ca

**Yuriko Oshima-Takane**
yuriko@psych.mcgill.ca

**Yoshio Takane**
takane@psych.mcgill.ca

Department of Psychology
McGill University
Montréal, Québec, Canada H3A 1B1

## Abstract

Understanding knowledge representations in neural nets has been a difficult problem. Principal components analysis (PCA) of contributions (products of sending activations and connection weights) has yielded valuable insights into knowledge representations, but much of this work has focused on the correlation matrix of contributions. The present work shows that analyzing the variance-covariance matrix of contributions yields more valid insights by taking account of weights.

## 1 INTRODUCTION

The knowledge representations learned by neural networks are usually difficult to understand because of the non-linear properties of these nets and the fact that knowledge is often distributed across many units. Standard network analysis techniques, based on a network's connection weights or on its hidden unit activations, have been limited. Weight diagrams are typically complex and weights vary across multiple networks trained on the same problem. Analysis of activation patterns on hidden units is limited to nets with a single layer of hidden units without cross connections.

Cross connections are direct connections that bypass intervening hidden unit layers. They increase learning speed in static networks by focusing on linear relations (Lang & Witbrock, 1988) and are a standard feature of generative algorithms such as cascade-correlation (Fahlman & Lebiere, 1990). Because such cross connections do so much of the work, analyses that are restricted to hidden unit activations furnish only a partial picture of the network's knowledge.

Contribution analysis has been shown to be a useful technique for multi-layer, cross connected nets. Sanger (1989) defined a contribution as the product of an output weight, the activation of a sending unit, and the sign of the output target for that input. Such contributions are potentially more informative than either weights alone or hidden unit activations alone since they take account of both weight and sending activation. Shultz and Elman (1994) used PCA to reduce the dimensionality of such contributions in several different types of cascade-correlation nets. Shultz and Oshima-Takane (1994) demonstrated that PCA of unscaled contributions produced even better insights into cascade-correlation solutions than did comparable analyses of contributions scaled by the sign of output targets. Sanger (1989) had recommended scaling contributions by the signs of output targets in order to determine whether the contributions helped or hindered the network's solution. But since the signs of output targets are only available to networks during error

correction learning, it is more natural to use unscaled contributions in analyzing knowledge representations.

There is an issue in PCA about whether to use the correlation matrix or the variance-covariance matrix. The correlation matrix contains 1s in the diagonal and Pearson correlation coefficients between contributions off the diagonal. This has the effect of standardizing the variables (contributions) so that each has a mean of 0 and standard deviation of 1. Effectively, this ensures that the PCA of a correlation matrix exploits variation in input activation patterns but ignores variation in connection weights (because variation in connection weights is eliminated as the contributions are standardized).

Here, we report on work that investigates whether more useful insights into network knowledge structures can be revealed by PCA of unstandardized contributions. To do this, we apply PCA to the variance-covariance matrix of contributions. The variance-covariance matrix has contribution variances along the diagonal and covariances between contributions off the diagonal. Taking explicit account of the variation in connection weights in this way may produce a more valid picture of the network's knowledge.

We use some of the same networks and problems employed in our earlier work (Shultz & Elman, 1994; Shultz & Oshima-Takane, 1994) to facilitate comparison of results. The problems include continuous XOR, arithmetic comparisons involving addition and multiplication, and distinguishing between two interlocking spirals. All of the nets were generated with the cascade-correlation algorithm (Fahlman & Lebiere, 1990).

Cascade-correlation begins as a perceptron and recruits hidden units into the network as it needs them in order to reduce error. The recruited hidden unit is the one whose activations correlate best with the network's current error. Recruited units are installed in a cascade, each on a separate layer and receiving input from the input units and from any previously existing hidden units. We used the default values for all cascade-correlation parameters.

The goal of understanding knowledge representations learned by networks ought to be useful in a variety of contexts. One such context is cognitive modeling, where the ability of nets to merely simulate psychological phenomena is not sufficient (McCloskey, 1991). In addition, it is important to determine whether the network representations bear any systematic relation to the representations employed by human subjects .

## 2   PCA OF CONTRIBUTIONS

Sanger's (1989) original contribution analysis began with a three-dimensional array of contributions (output unit x hidden unit x input pattern). In contrast, we start with a two-dimensional output weight x input pattern array of contributions. This is more efficient than the slicing technique used by Sanger to focus on particular output or hidden units and still allows identification of the roles of specific contributions (Shultz & Elman, 1994; Shultz & Oshima-Takane, 1994).

We subject the variance-covariance matrix of contributions to PCA in order to identify the main dimensions of variation in the contributions (Jolliffe, 1986). A component is a line of best fit to a set of data points in multi-dimensional space. The goal of PCA is to summarize a multivariate data set with a relatively small number of components by capitalizing on covariance among the variables (in this case, contributions).

We use the scree test (Cattell, 1966) to determine how many components are useful to include in the analysis. Varimax rotation is applied to improve the interpretability of the solution. Component scores are plotted to identify the function of each component.

## 3   APPLICATION TO CONTINUOUS XOR

The classical binary XOR problem does not have enough training patterns to make contribution analysis worthwhile. However, we constructed a continuous version of the XOR problem by dividing the input space into four quadrants. Starting from 0.1, input values were incremented in steps of 0.1, producing 100 $x, y$ input pairs that can be partitioned into four quadrants of the input space. Quadrant $a$ had values of $x$ less than

0.55 combined with values of *y* above 0.55. Quadrant *b* had values of *x* and *y* greater than 0.55. Quadrant *c* had values of *x* and *y* less than 0.55. Quadrant *d* had values of *x* greater than 0.55 combined with values of *y* below 0.55. Similar to binary XOR, problems from quadrants *a* and *d* had a positive output target (0.5) for the net, whereas problems from quadrants *b* and *c* had a negative output target (-0.5). There was a single output unit with a sigmoid activation.

Three cascade-correlation nets were trained on continuous XOR. Each of these nets generated a unique solution, recruiting five or six hidden units and taking from 541 to 765 epochs to learn to correctly classify all of the input patterns. Generalization to test patterns not in the training set was excellent. PCA of unscaled, unstandardized contributions yielded three components. A plot of rotated component scores for the 100 training patterns of net 1 is shown in Figure 1. The component scores are labeled according to their respective quadrant in the input space. Three components are required to account for 96.0% of the variance in the contributions.

Figure 1 shows that component 1, with 44.3% of the variance in contributions, has the role of distinguishing those quadrants with a positive output target (*a* and *d*) from those with a negative output target (*b* and *c*). This is indicated by the fact that the black shapes are at the top of the component space cube in Figure 1 and the white shapes are at the bottom. Components 2 and 3 represent variation along the *x* and *y* input dimensions, respectively. Component 2 accounted for 26.1% of the variance in contributions, and component 3 accounted for 25.6% of the variance in contributions. Input pairs from quadrants *b* and *d* (square shapes) are concentrated on the negative end of component 2, whereas input pairs from quadrants *a* and *c* (circle shapes) are concentrated on the positive end of component 2. Similarly, input pairs from quadrants *a* and *b* cluster on the negative end of component 3, and input pairs from quadrants *c* and *d* cluster on the positive end of component 3. Although the network was not explicitly trained to represent the *x* and *y* input dimensions, it did so as an incidental feature of its learning the distinction between quadrants *a* and *d* vs. quadrants *b* and *c*. Similar results were obtained from the other two nets learning the continuous XOR problem.

In contrast, PCA of the correlation matrix from these nets had yielded a somewhat less clear picture with the third component separating quadrants *a* and *d* from quadrants *b* and *c*, and the first two components representing variation along the *x* and *y* input dimensions (Shultz & Oshima-Takane, 1994). PCA of the correlation matrix of scaled contributions had performed even worse, with plots of component scores indicating interactive separation of the four quadrants, but with no clear roles for the individual components (Shultz & Elman, 1994).

Standardized, rotated component loadings for net 1 are plotted in Figure 2. Such plots can be examined to determine the role played by each contribution in the network. For example, hidden units 2, 3, and 4 all play a major role in the job done by component 1, distinguishing positive from negative outputs.

## 4  APPLICATION TO COMPARATIVE ARITHMETIC

Arithmetic comparison requires a net to conclude whether a sum or a product of two integers is greater than, less than, or equal to a comparison integer. Several psychological simulations have used neural nets to make additive and multiplicative comparisons and this has enhanced interest in this type of problem (McClelland, 1989; Shultz, Schmidt, Buckingham, & Mareschal, in press).

The first input unit coded the type of arithmetic operation to be performed: 0 for addition and 1 for multiplication. Three additional linear input units encoded the integers. Two of these input units each coded a randomly selected integer in the range of 0 to 9, inclusive; another input unit coded a randomly selected comparison integer. For addition problems, comparison integers ranged from 0 to 19, inclusive; for multiplication, comparison integers ranged from 0 to 82, inclusive. Two sigmoid output units coded the results of the comparison operation. Target outputs of 0.5, -0.5 represented a *greater than* result, targets of -0.5, 0.5 represented *less than*, and targets of 0.5, 0.5 represented *equal to*.

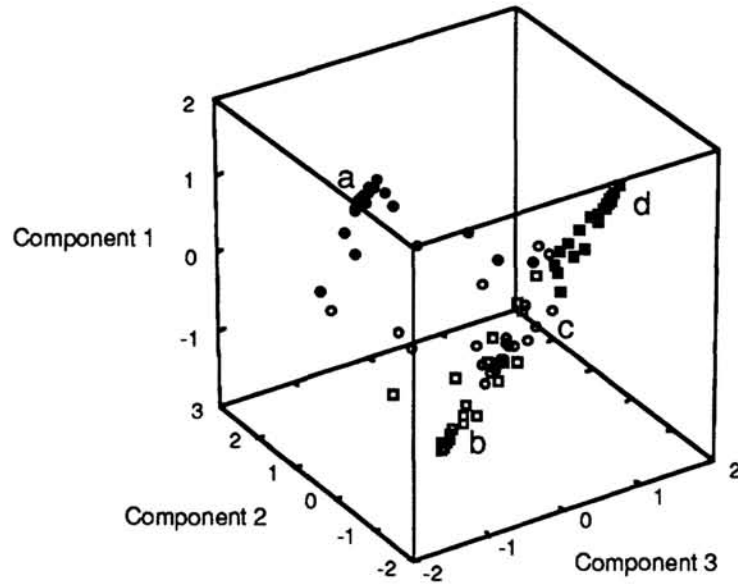

Figure 1. Rotated component scores for a continuous XOR net. Component scores for the *x, y* input pairs in quadrant *a* are labeled with black circles, those from quadrant *b* with white squares, those from quadrant *c* with white circles, and those from quadrant *d* with black squares. The network's task is to distinguish pairs from quadrants *a* and *d* (the black shapes) from pairs from quadrants *b* and *c* (the white shapes). Some of the white shapes appear black because they are so densely packed, but all of the truly black shapes are relatively high in the cube.

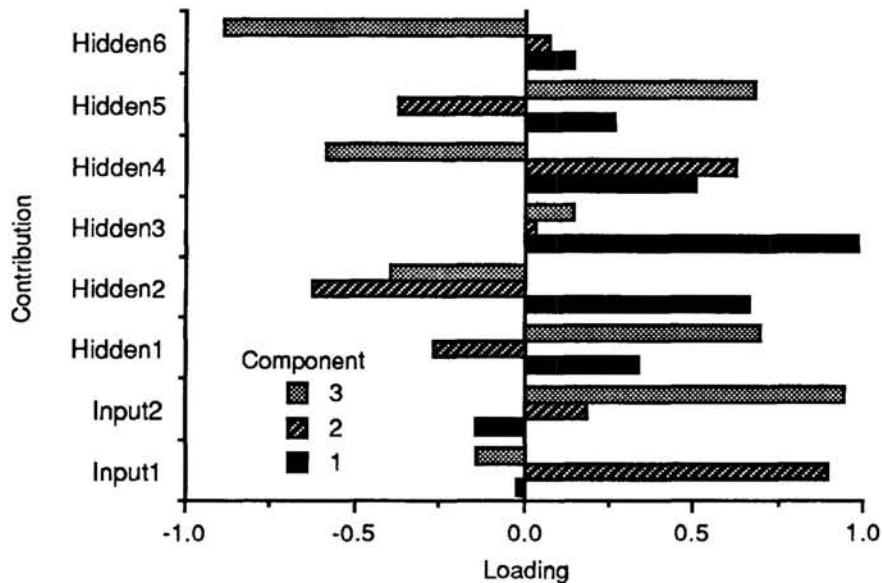

Figure 2. Standardized, rotated component loadings for a continuous XOR net. Rotated loadings were standardized by dividing them by the standard deviation of the respective contribution scores.

The training patterns had 100 addition and 100 multiplication problems, randomly selected, with the restriction that 45 of each had correct answers of *greater than*, 45 of each had correct answers of *less than*, and 10 of each had correct answers of *equal to*. These constraints were designed to reduce the natural skew of comparative values in the high direction on multiplication problems.

We ran three nets for 1000 epochs each, at which point they were very close to mastering the training patterns. Either seven or eight hidden units were recruited along the way. Generalization to previously unseen test problems was very accurate. Four components were sufficient to account for most the variance in unstandardized contributions, 88.9% in the case of net 1.

Figure 3 displays the rotated component scores for the first two components of net 1. Component 1, accounting for 51.1% of the variance, separated problems with *greater than* answers from problems with *less than* answers, and located problems with *equal to* answers in the middle, at least for addition problems. Component 2, with 20.2% of the variance, clearly separated multiplication from addition. Contributions from the first input unit were strongly associated with component 2. Similar results obtained for the other two nets.

Components 3 and 4, with 10.6% and 7.0% of the variance, were sensitive to variation in the second and third inputs, respectively. This is supported by an examination of the mean input values of the 20 most extreme component scores on these two components. Recall that the second and third inputs coded the two integers to be added or multiplied. The negative end of component 3 had a mean second input value of 8.25; the positive end of this component had a mean second input value of 0.55. Component 4 had mean third input value of 2.00 on the negative end and 7.55 on the positive end.

In contrast, PCA of the correlation matrix for these nets had yielded a far more clouded picture, with the largest components focusing on input variation and lesser components doing bits and pieces of the separation of answer types and operations in an interactive manner (Shultz & Oshima-Takane, 1994). Problems with *equal to* answers were not isolated by any of the components. PCA of scaled contributions had produced three components that interactively separated the three answer types and operations, but failed to represent variation in input integers (Shultz & Elman, 1994). Essentially similar advantages for using the variance-covariance matrix were found for nets learning either addition alone or multiplication alone.

# 5 APPLICATION TO THE TWO-SPIRALS PROBLEM

The two-spirals problem requires a particularly difficult discrimination and a large number of hidden units. The input space is defined by two interlocking spirals that wrap around their origin three times. There are two sets of 97 real-valued $x, y$ pairs, with each set representing one of the spirals, and a single sigmoid output unit coded for the identity of the spiral. Our three nets took between 1313 and 1723 epochs to master the distinction, and recruited from 12 to 16 hidden units. All three nets generalized well to previously unseen input pairs on the paths of the two spirals.

PCA of the variance-covariance matrix for net 1 revealed that six components accounted for a total of 97.9% of the variance in contributions. The second and fourth of these components together distinguished one spiral from the other, with 20.7% and 9.8% of the variance respectively. Rotated component scores for these two components are plotted in Figure 4. A diagonal line drawn on Figure 4 from coordinates -2, 2 to 2, -2 indicates that 11 points from each spiral were misclassified by components 2 and 4. This is only 11.3% of the data points in the training patterns. The fact that the net learned all of the training patterns implies that these exceptions were picked up by other components.

Components 1 and 6, with 40.7% and 6.4% of the variance, were sensitive to variation in the $x$ and $y$ inputs, respectively. Again, this was confirmed by the mean input values of the 20 most extreme component scores on these two components. On component 1, the negative end had a mean $x$ value of 3.55 and the positive end had a mean $y$ value of -3.55.

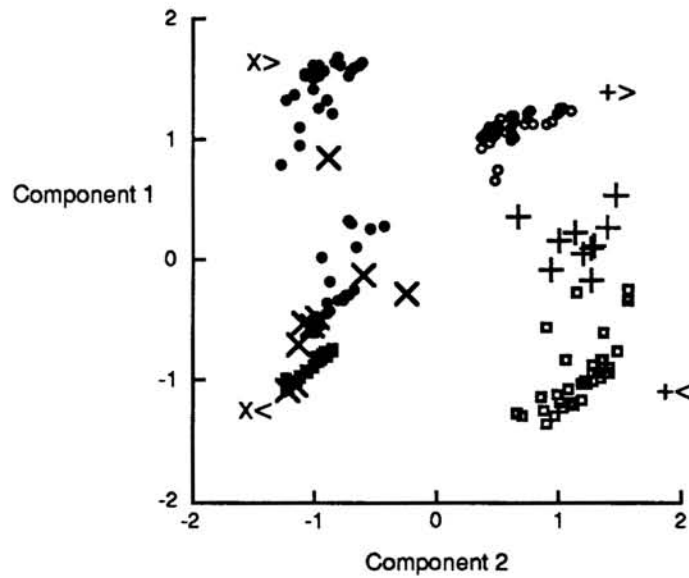

Figure 3. Rotated component scores for an arithmetic comparison net. *Greater than* problems are symbolized by circles, *less than* problems by squares, addition by white shapes, and multiplication by black shapes. For *equal to* problems only, addition is represented by + and multiplication by X. Although some densely packed white shapes may appear black, they have no overlap with truly black shapes. All of the black squares are concentrated around coordinates -1, -1.

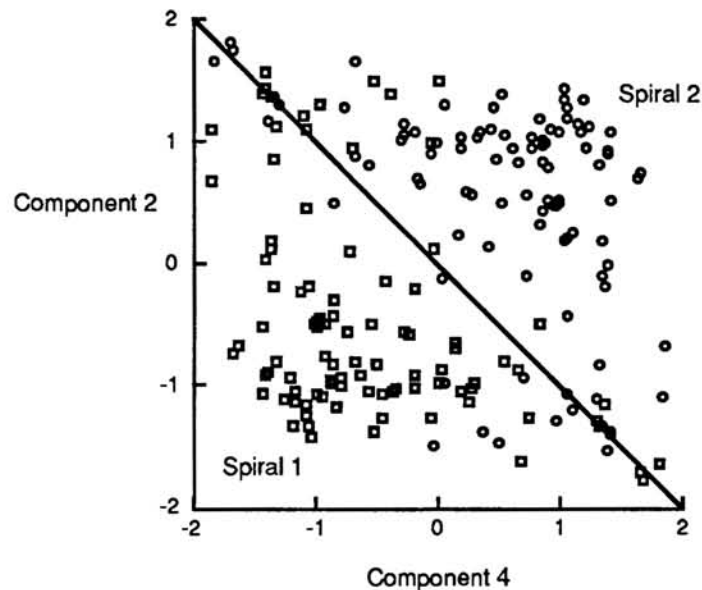

Figure 4. Rotated component scores for a two-spirals net. Squares represent data points from spiral 1, and circles represent data points from spiral 2.

On component 6, the negative end had a mean $x$ value of 2.75 and the positive end had a mean $y$ value of -2.75. The skew-symmetry of these means is indicative of the perfectly symmetrical representations that cascade-correlation nets achieve on this highly symmetrical problem. Every data point on every component has a mirror image negative with the opposite signed component score on that same component. This $-x$, $-y$ mirror image point is always on the other spiral. Other components concentrated on particular regions of the spirals. The other two nets yielded essentially similar results.

These results can be contrasted with our previous analyses of the two-spirals problem, none of which succeeded in showing a clear separation of the two spirals. PCAs based on scaled (Shultz & Elman, 1994) or unscaled (Shultz & Oshima-Takane, 1994) correlation matrices showed extensive symmetries but never a distinction between one spiral and another.[1] Thus, although it was clear that the nets had encoded the problem's inherent symmetries, it was still unclear from previous work how the nets used this or other information to distinguish points on one spiral from points on the other spiral.

## 6 DISCUSSION

On each of these problems, there was considerable variation among network solutions, as revealed, for example, by variation in numbers of hidden units recruited and signs and sizes of connection weights. In spite of such variation, the present technique of applying PCA to the variance-covariance matrix of contributions yielded results that are sufficiently abstract to characterize different nets learning the same problem. The knowledge representations produced by this analysis clearly identify the essential information that the net is being trained to utilize as well as more incidental features of the training patterns such as the nature of the input space.

This research strengthens earlier conclusions that PCA of network contributions is a useful technique for understanding network performance (Sanger, 1989), including relatively intractable multi-level cross connected nets (Shultz & Elman, 1994; Shultz & Oshima-Takane, 1994). However, the current study underscores the point that there are several ways to prepare a contribution matrix for PCA, not all of which yield equally valid or useful results. Rather than starting with a three dimensional matrix of output unit x hidden unit x input pattern and focusing on either one output unit at a time or one hidden unit at a time (Sanger, 1989), it is preferable to collapse contributions into a two dimensional matrix of output weight x input pattern. The latter is not only more efficient, but yields more valid results that characterize the network as a whole, rather than small parts of the network.

Also, rather than scaling contributions by the sign of the output target (Sanger, 1989), it is better to use unscaled contributions. Unscaled contributions are not only more realistic, since the network has no knowledge of output targets during its feed-forward phase, but also produce clearer interpretations of the net's knowledge representations (Shultz & Oshima-Takane, 1994). The latter claim is particularly true in terms of sensitivity to input dimensions and to operational distinctions between adding and multiplying. Plots of component scores based on unscaled contributions are typically not as dense as those based on scaled contributions but are more revealing of the network's knowledge.

Finally, rather than applying PCA to the correlation matrix of contributions, it makes more sense to apply it to the variance-covariance matrix. As noted in the introduction, using the correlation matrix effectively standardizes the contributions to have identical means and variances, thus obscuring the role of network connection weights. The present results indicate much clearer knowledge representations when the variance-covariance matrix is used since connection weight information is explicitly retained. Matrix differences were especially marked on the more difficult problems, such as two-spirals, where the only PCAs to reveal how nets distinguished the spirals were those based on

variance-covariance matrices. But the relative advantages of using the variance-covariance matrix were evident on the easier problems too.

There has been recent rapid progress in the study of the knowledge representations learned by neural nets. Feed-forward nets can be viewed as function approximators for relating inputs to outputs. Analysis of their knowledge representations should reveal how inputs are encoded and transformed to produce the correct outputs. PCA of network contributions sheds light on how these function approximations are done. Components emerging from PCA are orthonormalized ingredients of the transformations of inputs that produce the correct outputs. Thus, PCA helps to identify the nature of the required transformations.

Further progress might be expected from combining PCA with other matrix decomposition techniques. Constrained PCA uses external information to decompose multivariate data matrices before applying PCA (Takane & Shibayama, 1991).

Analysis techniques emerging from this research will be useful in understanding and applying neural net research. Component loadings, for example, could be used to predict the results of lesioning experiments with neural nets. Once the role of a hidden unit has been identified by virtue of its association with a particular component, then one could predict that lesioning this unit would impair the function served by the component.

### Acknowledgments

This research was supported by the Natural Sciences and Engineering Research Council of Canada.

## Footnotes

[1]Results from unscaled contributions on the two-spirals problem were not actually presented in Shultz & Oshima-Takane (1994) since they were not very clear.

### References

Cattell, R. B. (1966). The scree test for the number of factors. *Multivariate Behavioral Research*, 1, 245-276.

Fahlman, S. E., & Lebiere, C. (1990.) The Cascade-Correlation learning architecture. In D. Touretzky (Ed.), *Advances in neural information processing systems 2*, (pp. 524-532). Mountain View, CA: Morgan Kaufmann.

Jolliffe, I. T. (1986). *Principal component analysis*. Berlin: Springer Verlag.

Lang, K. J., & Witbrock, M. J. (1988). Learning to tell two spirals apart. In D. Touretzky, G. Hinton, & T. Sejnowski (Eds)., *Proceedings of the Connectionist Models Summer School*, (pp. 52-59). Mountain View, CA: Morgan Kaufmann.

McClelland, J. L. (1989). Parallel distributed processing: Implications for cognition and development. In Morris, R. G. M. (Ed.), *Parallel distributed processing: Implications for psychology and neurobiology*, pp. 8-45. Oxford University Press.

McCloskey, M. (1991). Networks and theories: The place of connectionism in cognitive science. *Psychological Science*, 2, 387-395.

Sanger, D. (1989). Contribution analysis: A technique for assigning responsibilities to hidden units in connectionist networks. *Connection Science*, 1, 115-138.

Shultz, T. R., & Elman, J. L. (1994). Analyzing cross connected networks. In J. D. Cowan, G. Tesauro, & J. Alspector (Eds.), *Advances in Neural Information Processing Systems 6*. San Francisco, CA: Morgan Kaufmann.

Shultz, T. R., & Oshima-Takane, Y. (1994). Analysis of unscaled contributions in cross connected networks. In *Proceedings of the World Congress on Neural Networks* (Vol. 3, pp. 690-695). Hillsdale, NJ: Lawrence Erlbaum.

Shultz, T. R., Schmidt, W. C., Buckingham, D., & Mareschal, D. (In press). Modeling cognitive development with a generative connectionist algorithm. In G. Halford & T. Simon (Eds.), *Developing cognitive competence: New approaches to process modeling*. Hillsdale, NJ: Erlbaum.

Takane, Y., & Shibayama, T. (1991). Principal component analysis with external information on both subjects and variables. *Psychometrika*, 56, 97-120.
